# Shape-Based Object Localization
# for Descriptive Classification

**Geremy Heitz**[1,*]     **Gal Elidan**[2,3,*]     **Ben Packer**[2,*]     **Daphne Koller**[2]

[1]Department of Electrical Engineering, Stanford University
[2]Department of Computer Science, Stanford University
[3]Department of Statistics, Hebrew University, Jerusalem

{gaheitz,bpacker,koller}@cs.stanford.edu   galel@huji.ac.il

## Abstract

Discriminative tasks, including object categorization and detection, are central components of high-level computer vision. Sometimes, however, we are interested in more refined aspects of the object in an image, such as pose or particular regions. In this paper we develop a method (LOOPS) for learning a shape and image feature model that can be trained on a particular object class, and used to outline instances of the class in novel images. Furthermore, while the training data consists of uncorresponded outlines, the resulting LOOPS model contains a set of landmark points that appear consistently across instances, and can be accurately localized in an image. Our model achieves state-of-the-art results in precisely outlining objects that exhibit large deformations and articulations in cluttered natural images. These localizations can then be used to address a range of tasks, including descriptive classification, search, and clustering.

## 1   Introduction

Discriminative questions such as "What is it?" (categorization) and "Where is it?" (detection) are central to machine vision and have received much attention in recent years. In many cases, we are also interested in more refined descriptive questions with regards to an object such as "What is it doing?", "What is its pose?", or "What color is its tail?". For example, we may wish to determine whether a cheetah is running, or whether a giraffe is bending over to drink. In a shopping scenario, we might be interested in searching for lamps that have a particular type of lampshade.

In theory it is possible to convert some descriptive questions into discriminative classification tasks given the appropriate labels. Nevertheless, it is preferable to have a single framework in which we can answer a range of questions, some of which may not be known at training time, or may not be discriminative in nature. Intuitively, if we have a good model of what objects in a particular class "look like" and the range of variation that they exhibit, we can make these descriptive distinctions more readily, with a small number of training instances. Furthermore, such a model allows us the flexibility to perform clustering, search, and other forms of exploration of the data.

In this paper, we address the goal of finding *precise*, *corresponded* localizations of object classes in cluttered images while allowing for large deformations. The **L**ocalizing **O**bject **O**utlines using **P**robabilistic **S**hape (LOOPS) method constructs a unified probabilistic model that combines global shape with appearance-based boosted detectors to define a joint distribution over the location of the constituent elements on the object. We can then leverage the object's shape, an important characteristic that can be used for many descriptive distinctions [9], to address our descriptive tasks. The main challenge is to correspond this model to a novel image while accounting for the possibility of object deformation and articulation.

Contour-based methods such as active shape/appearance models (AAMs) [4] were developed with this goal in mind, but typically require good initial guesses and are applied to images with significantly less clutter than real-life photographs. As a result, AAMs have not been successfully used for

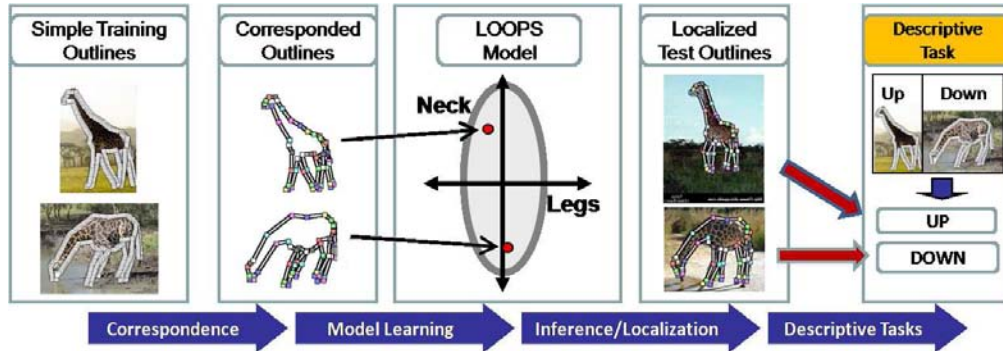

Figure 1: The stages of LOOPS. The shape model is depicted via principal components corresponding to the neck and legs, and the ellipse marks one standard deviation from the mean. Red circles show the location of sample instances in this space. Descriptive tasks other than classification (right box) are described in Section 4.

class-level object recognition/analysis. Some works use geometry as a means toward object classification or detection [11, 2, 17, 21]). Since, for example, a misplaced leg has a negligible effect on classification, these works do not attempt to optimize localization. Other works (e.g., [3, 12]) do attempt to accurately localize objects in photographs but only allow for relatively rigid configurations, and cannot capture large deformations such as the articulation of the giraffe's neck. To the best of our knowledge, no work uses the consistent localization of parts for descriptive tasks.

Having a representation of the constituent elements of an object should aid in answering descriptive questions. For example, to decide whether a giraffe is standing upright or bending down to drink, we can use a specific representation of the head, neck, body, and legs in order to consider their relative location. We adopt the AAM-like strategy of representing the shape of an object class via an ordered set of $N$ landmark points that together constitute a piecewise linear contour.

Obtaining corresponded training outlines, however, requires painstaking supervision and we would like to be able to use readily available simple outlines such as those in the LabelMe dataset. Therefore, before we begin, we need to *automatically* augment the simple training outlines with a corresponded labeling. That is, we want to transform arbitrary outlines into useful training instances with consistent elements as depicted in the pipeline of our LOOPS method (Figure 1, first two boxes). The method we use for this step is reminiscent of Hill and Taylor [14]; we omit the details for lack of space. Once we have corresponded training outlines, each with $N$ consistent landmarks, we can construct a distribution of the geometry of the objects' outline as depicted in Figure 1(middle) and augment this with appearance based features to form a LOOPS model, as described in Section 2.

Given a model, we face the computational challenge of localizing the landmarks in test images in the face of clutter, large deformations, and articulations (Figure 1, fourth box). In order to overcome the problem of local maxima faced by contour propagation methods (e.g., [4, 20]), we develop a two-stage scheme. We first consider a tractable global search space, consisting of candidate landmark assignments. This allows a discrete probabilistic inference technique to achieve rough but accurate localization that robustly explores the multimodal set of solutions allowed by our large deformation model. We then refine our localization using a continuous hill-climbing approach. This hybrid approach allows LOOPS to deal effectively with complex images of natural scenes, without requiring a good initialization. Preliminary investigations showed that a simpler approach that does a purely local search, similar to the AAMs of Cootes et al. [4], was unable to deal with the challenges of our data. The localization of outlines in test images is described in detail in Section 3. We demonstrate in Section 4 that this localization achieves state-of-the-art results for objects with significant deformation and articulation in natural images.

Finally, with the localized outlines in hand, we can readily perform a range of descriptive tasks (classification, ranking, clustering), based on the predicted location of landmarks in test images as well as appearance characteristics in the vicinity of those landmarks. We demonstrate how this is carried out for several descriptive tasks in Section 4. We explore the space of applications facilitated by the LOOPS model across two principal axes. The first concerns the machine learning application: we present results for classification, search (ranking), and clustering. The second axis varies the components that are extracted from the LOOPS outlines for these tasks: we show examples that use the entire object shape, a subcomponent of the object shape, and the appearance of a specific part of the object. The LOOPS framework allows us to approach any of these tasks with a single model without the need for retraining.

## 2 The LOOPS Model

Given a set of training instances, each with $N$ corresponded landmarks, the LOOPS object class model combines two components: an explicit representation of the object's shape (2D silhouette), and a set of image-based features. We define the shape of a class of objects via the locations of the $N$ object landmarks, each of which is assigned to one of the image pixels. We represent such an assignment as a $2N$ vector of image coordinates which we denote by $\mathbf{L}$. Using the language of Markov random fields [18], the LOOPS model defines a conditional probability distribution over $\mathbf{L}$:

$$P(\mathbf{L} \mid \mathcal{I}, \Theta) = \frac{1}{Z(\mathcal{I})} P_{\text{Shape}}(\mathbf{L}; \mu, \Sigma) \prod_i \exp\left(w_i F_i^{\text{det}}(l_i; \mathcal{I})\right) \prod_{i,j} \exp\left(w_{ij} F_{ij}^{\text{grad}}(l_i, l_j; \mathcal{I})\right) \quad (1)$$

where $\Theta = \{\mu, \Sigma, \mathbf{w}\}$ are the model parameters, and $i$ and $j$ index the model landmarks. $P_{\text{Shape}}$ encodes the (unnormalized) distribution over the object shape (outline), $F^{\text{det}}(l_i)$ is a landmark specific detector, and $F_{ij}^{\text{grad}}(l_i, l_j; \mathcal{I})$ encodes a preference for aligning outline segments along image edges.

Below we describe how the shape model and the detector features are learned. We found that our results are quite robust to the choice of weights and that learning them provides no clear benefit. We note that our MRF formulation is quite general, and allows for both the incorporation of (possibly weighted) additional features. For instance, we might want to capture the notion that internal line segments (lines entirely contained within the object) should have low color variability. This can naturally be posed as a pairwise feature over landmarks on opposite sides of the object.

We model the shape component of Eq. (1) as a multivariate Gaussian distribution over landmark locations with mean $\mu$ and covariance $\Sigma$. The Gaussian parametric form has many attractive properties, and has been used successfully to model shape distributions in a variety of applications (e.g., [4, 1]). In our context, one particularly useful property is that the Gaussian distribution decomposes into a product of quadratic terms over pairs of variables:

$$P_{\text{Shape}}(\mathbf{L} \mid \mu, \boldsymbol{\Sigma}) = \frac{1}{Z} \prod_{i,j} \exp\left(-\frac{1}{2}(x_i - \mu_i)\boldsymbol{\Sigma}_{ij}^{-1}(x_j - \mu_j)\right) = \frac{1}{Z} \prod_{i,j} \phi_{i,j}(x_i, x_j; \mu, \boldsymbol{\Sigma}),$$

where $Z$ is the normalization term. As this equation illustrates, we can specify potentials $\phi_{i,j}$ over only singletons and pairs of variables and still manage to represent the full shape distribution. This allows Eq. (1) to take an appealing form in which all terms are defined over at most a two variables.

As we discuss below in Section 3, the procedure to locate the model landmarks in an image first involves discrete global inference using the LOOPS model, followed by a local refinement stage. Even if we limit ourselves to pairwise terms, performing discrete inference in a densely connected MRF may be computationally impractical. Unfortunately, a general multivariate Gaussian includes pairwise terms between all landmarks. Thus, during the discrete inference stage, we limit the number of pairwise elements by approximating the shape distribution with a sparse multivariate Gaussian. (During the final refinement stage, we use the full distribution.) To obtain the sparsity pattern, we choose a linear number of landmark pairs whose relative locations have the lowest variance across the training instances (and require that neighbor pairs be included), promoting shape stability. The sparse Gaussian is then obtained by using a gradient method to minimize the KL distance to the full distribution subject to the entries corresponding to the chosen pairs being 0.

To construct detector features $F^{\text{det}}$, we build on the success of boosting in state-of-the-art object detection methods [17, 22]. Specifically, we use boosting to learn a strong detector (classifier), $H_i$ for each landmark $i$. We then define the feature value in the conditional MRF for the assignment of landmark $i$ to pixel $l_i$ to be $F_i^{\text{det}}(l_i; \mathcal{I}) = H_i(l_i)$.

For weak detectors we use features that are based on our shape model as well as other features that have proven useful for the task of object detection: shape templates [5], boundary fragments [17], filter response patches [22], and SIFT descriptors [16]. The weak detector $h_i^t(l_i)$ is one of these features chosen at round $t$ of boosting that best predicts whether landmark $i$ is at a particular pixel $l_i$. Boosting yields a strong detector of the form $H_i(l_i) = \sum_{t=1}^{T} \alpha_t h_i^t(l_i)$.

The pairwise feature $F_{ij}^{\text{grad}}(l_i, l_j; \mathcal{I}) = \sum_{r \in \overline{l_i l_j}} |\mathbf{g}(r)^T \mathbf{n}(l_i, l_j)|$ sums over the segment between adjacent landmarks, where $\mathbf{g}(r)$ is the image gradient at point $r$, and $\mathbf{n}(l_i, l_j)$ is the segment normal.

Figure 2: Example outlines predicted using (candidate) the top detection for each landmark independently, (discrete) inference, (c) a continuous refinement of (b).

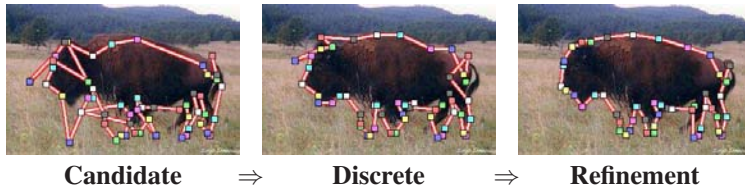

**Candidate** ⇒ **Discrete** ⇒ **Refinement**

## 3   Localization of Object Outlines

We now address our central computational challenge: assigning the landmarks of a LOOPS model to test image pixels while allowing for large deformations and articulations. Recall that the conditional MRF defines a distribution (Eq. (1)) over assignments of model landmarks to pixels. This allows us to outline objects by using probabilistic inference to find the most probable such assignment:

$$\mathbf{L}^* = \mathrm{argmax}_{\mathbf{L}} P(\mathbf{L} \mid \mathcal{I}, \mathbf{w})$$

Because, in principle, each landmark can be assigned to any pixel, finding $\mathbf{L}^*$ is computationally prohibitive. One option is to use an approach analogous to active shape models, using a greedy method to deform the model from a fixed starting point. However, unlike most applications of active shape/appearance models (e.g., [4]), our images have significant clutter, and such an approach will quickly get trapped in an inferior local maxima. A possible solution to this problem is to consider a series of starting points. Preliminary experiments along these lines (not shown for lack of space), however, showed that such an approach requires a computationally prohibitive number of starting points to effectively localize even rigid objects. Furthermore, large articulations were not captured even with the "correct" starting point (placing the mean shape in the center of the true location). To overcome these limitations, we propose an alternative two step method, depicted in Figure 2: we first approximate our problem and find a coarse solution using discrete inference; we then refine our solution using continuous optimization and the full objective defined by Eq. (1).

We cannot directly perform inference over the entire seach space of $N^P$ assigments (for $N$ model landmarks and $P$ pixels). To prune this space, we first assume that landmarks will fall on "interesting" points, and consider only candidate pixels (typically 1000-2000 per image) found by the SIFT interest operator [16]. We then use the appearance based features $F_i^{\mathrm{det}}$ to rank the pixel candidates and choose the top $K$ (25) candidate pixels for each landmark. Even with this pruned space, the inference problem is quite daunting, so we further approximate our objective by sparsifying the multivariate Gaussian shape distribution, as mentioned in Section 2. The only pairwise feature functions we use are over neighboring pairs of landmarks (as described in Section 2), which does not add to the density of the MRF construction, thus allowing the inference procedure to be tractable. We perform approximate max-product inference using the Residual Belief Propagation (RBP) algorithm [6] to find the most likely assignment of landmarks to pixels $\mathbf{L}^*$ in the pruned space.

Given the best assignment $\mathbf{L}^*$ predicted in the discrete stage, we perform a refinement stage in which we reintroduce the entire pixel domain and use the full shape distribution. Refinement involves a greedy hill-climbing algorithm in which we iterate across each landmark, moving it to the best candidate location using one of two types of moves, while holding the other landmarks fixed. In a **local** move, each landmark picks the best pixel in a small window around its current location. In a **global** move, each landmark can move to its mean location given all the other landmark assignments; this location is the mean of the conditional Gaussian $P_{\mathrm{Shape}}(l_i \mid \mathbf{L} \setminus l_i)$, easily computed from the joint shape Gaussian. In a typical refinement, the **global** moves dominate the early iterations, correcting large mistakes made by the discrete stage and that resulted in an unlikely shape. In the later iterations, **local** moves do most of the work by carefully adapting to the local image characteristics.

## 4   Experimental Results

Our experimental evaluation is aimed at demonstrating the ability of a single LOOPS model to perform a range of tasks based on corresponded localization of objects. In the experiments in the following sections, we train on 20 instances of each class and test on the rest, and report results averaged over 5 random train/test partitions. For the "airplane" image class, we selected examples from the Caltech airplanes image set [8]; the other classes were gathered for this paper. More detailed results, including more object classes and scenes, and an analysis of outline accuracy, appear in [13]. Full image results appear at http://ai.stanford.edu/~gaheitz/Research/Loops.

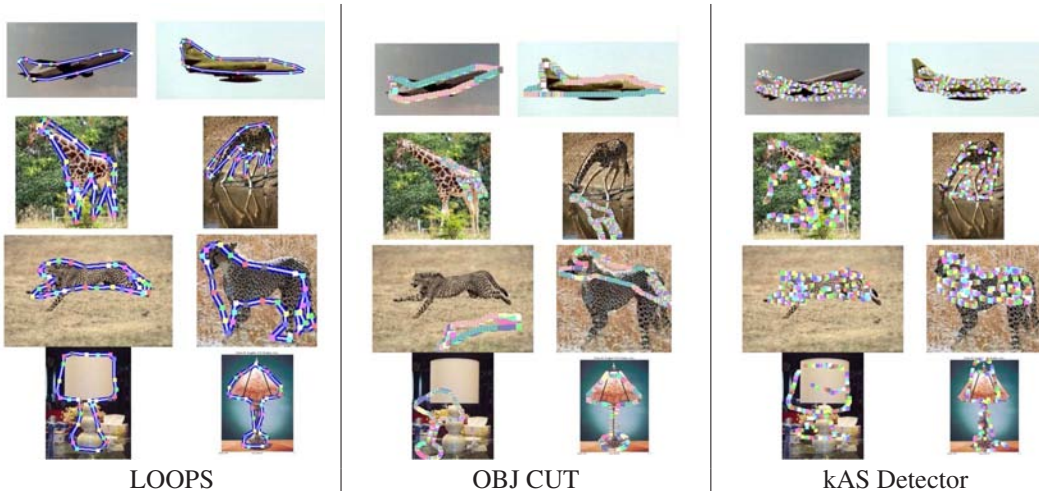

|  | LOOPS | OBJ CUT | kAS Detector |
|---|---|---|---|

Figure 3: Randomly selected outlines produced by LOOPS and its two competitors, displaying the variation in the four classes considered in our descriptive classification experiments.

| Class | LOOPS | OBJ CUT | kAS |
|---|---|---|---|
| Airplane | 2.0 | 6.0 | 3.9 |
| Cheetah | 5.2 | 12.7 | 11.9 |
| Giraffe | 2.9 | 11.7 | 8.9 |
| Lamp | 2.9 | 7.5 | 5.8 |

Table 1: Normalized symmetric root mean squared (rms) outline error. We report the rms of the distance from each point on the outline to the nearest point on the groundtruth (and vice versa), as a percentage of the groundtruth bounding box diagonal.

**Accurate Outline Localization**

In order for a LOOPS model to achieve its goals of classification, search and clustering based on characteristics of the shape or shape-localized appearance, it is necessary for our localization to be accurate at a more refined level than the bounding box prediction that is typical in the literature. We first evaluate the ability of our model to produce accurate outlines in which the model's landmarks are positioned consistently across test images.

We compare LOOPS to two state-of-the-art methods that seek to produce accurate object outlines in cluttered images: the **OBJ CUT** model of Prasad and Fitzgibbon [19] and the **kAS Detector** of Ferrari et al. [12]. Both methods were updated to fit our data with help from the authors (P. Kumar, V. Ferrari; personal communications). Unlike both OBJ CUT and LOOPS, the kAS Detector only requires bounding box supervision for the training images rather than full outlines. To provide a quantitative evaluation of the outlines, we measured the symmetric root mean squared (rms) distance between the produced outlines and the hand-labeled groundtruth. As we can see both qualitatively in Figure 3 and quantitatively in Table 1, LOOPS produces significantly more accurate outlines than its competitors. Figure 3 shows two example test images with the outlines for each of the four classes we considered here. While in some cases the LOOPS outline is not perfect at the pixel level, it usually captures the correct articulation, pose, and shape of the object.

**Descriptive Classification with LOOPS Outlines**

Our goal is to use the predicted LOOPS outlines for distinguishing between two configurations of an object. To accomplish this, we first train the joint shape and appearance model and perform inference to localize outlines in the test images, all *without* knowledge of the classification task or any labels. Representing each instance as a corresponded outline provides information that can be leveraged much more easily than the pixel-based representation. We then incorporate the labels to train a *descriptive* classifier given a corresponded localization.

To classify a test image, we used a nearest neighbor classifier, based on chamfer distance. The distance is computed efficiently by converting the training contours into an "ideal" edge image and computing the distance transform of this edge image. The **LOOPS** outlines are then classified based on their mean distance to each training contour. In addition, we include a **GROUND** measure that uses the landmark coordinates of manually corresponded groundtruth outlines as features in a logistic regression classifier. This serves as a rough upper bound on the performance achievable by

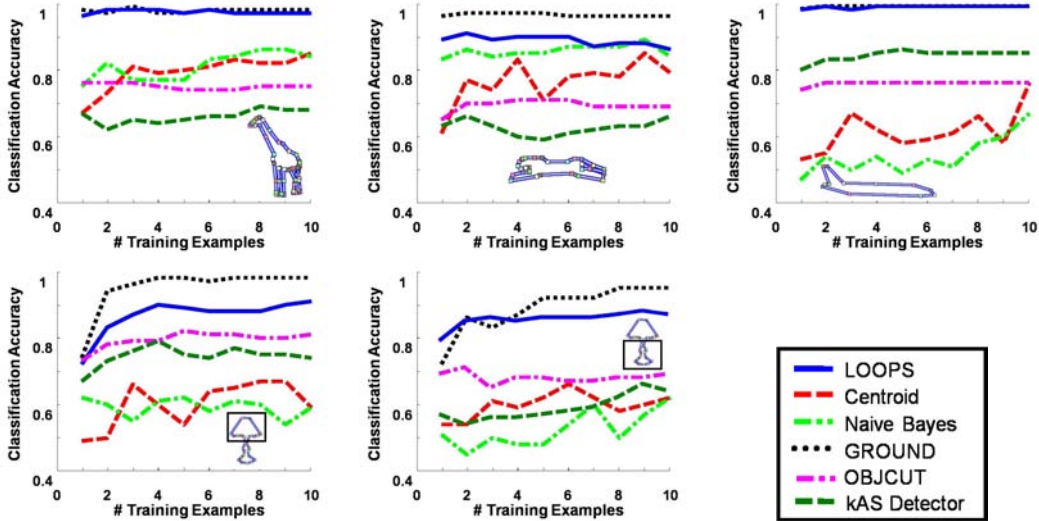

Figure 4: Descriptive classification results. **LOOPS** is compared to the **Naive Bayes** and boosted **Centroid** classifier baselines as well as the state-of-the-art **OBJ CUT** and **kAS Detector** methods. **GROUND** uses manually labeled outlines and approximately upper bounds the performance achievable from outlines. For both lamp tasks, the same LOOPS, OBJ CUT, and kAS Detector models and localizations are used. Note that unlike the other methods, the kAS Detector requires only bounding box supervision rather than full outlines.

relying on outlines. In practice, **LOOPS** can outperform **GROUND** if the classifier picks up on signals from the automatically chosen landmarks.

In addition to the kAS Detector and OBJ CUT competitors, we introduce to two baseline techniques for comparison. The first is a **Naive Bayes** classifier that uses a codebook of SIFT features as in [7]. The second uses a discriminative approach based on the **Centroid** detector described above, which is similar to the detector used by [22]; we train the descriptive classifier based on the vector of feature responses at the predicted object centroid.

Figure 4 (top) shows the classification results for three tasks: giraffes standing vs. bending down; cheetahs running vs. standing; and airplanes taking off vs. flying horizontally. The first two tasks depend on the *articulation* of the object, while the third depends on its *pose*. (In this last task, where rotation is the key feature, we only normalize for translation and scale when performing Procrustes alignment.) Classification performance is shown as a function of the number of labeled instances. For all three tasks, **LOOPS** (solid blue) outperforms both baselines as well as the state-of-the-art competitors. Importantly, by making use of the outline predicted in a cluttered image, we surpass the fully supervised baselines (rightmost on the graphs) with as little as a single supervised instance (leftmost on the graphs).

Once we have outlined instances, an important benefit of the LOOPS method is that we can in fact perform multiple descriptive tasks with the same object model. We demonstrate this with a pair of classification tasks for the lamp object class, presented in Figure 4(bottom). The tasks differ in which "part" of the object we consider for classification: triangular vs. rectangular lamp shade; and thin vs. fat lamp base. By including a few examples in the labeled set, our classifier can learn to consider only the relevant portion of the shape. We stress that both the learned lamp model and the test localizations predicted by LOOPS are the same for both tasks. Only the label set and the resulting nearest-neighbor classifier change. The consequences of this result are promising: we can do most of the work once, and then readily perform a range of descriptive classification tasks.

**Shape Similarity Search**
The second descriptive application area that we consider is similarity search, which involves the ranking of test instances based on their similarity to a search query. A shopping website, for example, might wish to allow a user to organize the examples in a database according to similarity to a query product. The similarity measure can be any feature that is easily extracted from the image while leveraging the predicted LOOPS outline. The experimental setup is as follows. Offline, we train a LOOPS model for the object class and localize corresponded outlines in the test images. On-

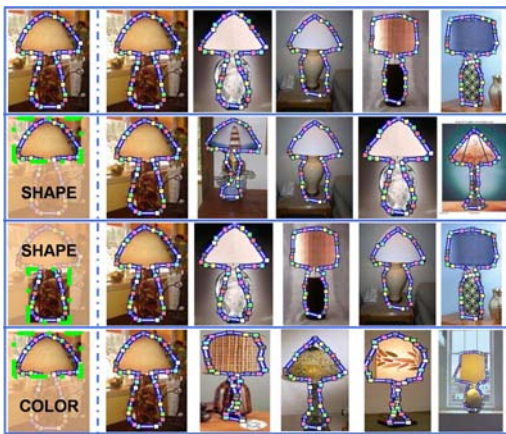

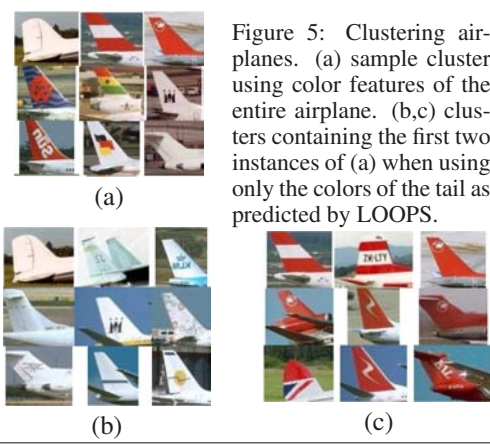

Figure 5: Clustering airplanes. (a) sample cluster using color features of the entire airplane. (b,c) clusters containing the first two instances of (a) when using only the colors of the tail as predicted by LOOPS.

(a)

(b)                                    (c)

Figure 6: (left) Object similarity search using the LOOPS output to determine the location of the lamp landmarks. (top row) searching the test database using full shape similarity to the query object on the left; (second row) evaluating similarity only using the landmarks that correspond to the lamp shade; (third row) search focused only on the lamp base; (bottom row) using color similarity of the lamp shade to rank the search results.

line, a user chooses a test instance to serve as a "query" image and a similarity metric to use. We search for the test images that are most similar to the query, and return the ranked list of images.

Figure 6 shows an example from the lamp dataset. Users select a query lamp instance, a subset of landmarks (possibly all), and whether to use shape or color. Each instance in the dataset is then ranked based on Euclidean distance to the query in shape PCA space or *LAB* color space as appropriate. The top row shows a full-shape search, where the left-most image is the query instance and the others are ordered by decreasing similarity. The second row shows the ranking when the user decides to focus on the lampshade landmarks, yielding only triangular lamp shades, and the third row focuses on the lamp base, returning only wide bases. Finally, the bottom row shows a search based on the color of the shade. In all of these examples, by projecting the images into LOOPS outlines, similarity search desiderata were easily specified and effectively taken into account. The similarity of interest in all of these cases is hard to specify without a predicted outline.

**Descriptive Clustering**

Finally, we consider clustering a database by leveraging on the LOOPS predicted outlines. As an example, we consider a large database of airplane images, and wish to group our images into "similar looking" sets of airplanes. Clustering based on shape might produce clusters corresponding to passenger jets, fighter jets, and small propeller airplanes. In this section, we consider an outline *and* appearance based clustering where the feature vector for each airplane includes the mean color values in the *LAB* color space for all pixels inside the airplane boundary (or in a region bounded by a user-selected set of landmarks). To cluster images based on this vector, we use standard K-means.

Figure 5(left) shows 12 examples from one cluster that results from clustering using the entire plane, for a database of 770 images from the Caltech airplanes image set [8]. Despite the fact that the cluster is coherent when considering the whole plane (not shown), zooming in on the tails reveals that the tails are quite heterogeneous in appearance. Figure 5(middle) and (right) show the tails for the two clusters that contain the first two instances from Figure 5(left), when using only the tail region for clustering. The coherence of the tail appearance is apparent in this case, and both clusters group many tails from the same airlines. In order to perform such coherent clustering of airplane tails, one needs first to accurately localize the tail in test images. Even more than the table lamp ranking task presented above, this example highlights the ability of LOOPS to leverage *localize appearance*, opening the door for many additional shape and appearance based descriptive tasks.

## 5   Discussion and Future Work

In this work we presented the **L**ocalizing **O**bject **O**utlines using **P**robabilistic **S**hape (LOOPS) approach for obtaining accurate, corresponded outlines of objects in test images, with the goal of performing a variety of descriptive tasks. Our approach relies on a coherent probabilistic model in which shape is combined with discriminative detectors. We showed how the produced outlines can

be used to perform descriptive classification, search, and clustering based on shape and localized appearance, and we evaluated the error of our outlines compared to two state-of-the-art competitors. For the classification tasks, we showed that our method is superior to fully supervised competitors with as little as a single labeled example.

Our contribution is threefold. First, we introduce a model that combines both generative and discriminative elements, allowing us to localize precise outlines of highly articulated objected in cluttered natural images. Second, in order to achieve this localization, we present a hybrid global-discrete then local-continuous optimization approach to the model-to-image correspondence problem. Third, we demonstrate that precise localization is of value for a range of descriptive tasks, including those that are based on appearance.

Several existing methods produce outlines either as a by-product of detection (e.g., [3, 17, 21]) or as a targeted goal (e.g., [12, 19]). In experiments above, we compared LOOPS to two state-of-the-art methods. We showed that LOOPS produces far more accurate outlines when dealing with significant object deformation and articulation, and demonstrated that it is able to translate this into superior classification rates for descriptive tasks. No other work that considers object classes in natural images has demonstrated a combination of accurate localization and shape analysis that has solved these problems.

There are further directions to pursue. We would like to automatically learn coherent parts of objects (e.g., the neck of the giraffe) as a set of landmarks that articulate together, and achieve better localization by estimating a distribution over part articulation (e.g., synchronized legs). A natural extension of our model is a scene-level variant in which each object is treated as a "landmark." The geometry of such a model will then capture relative spatial location and orientations so that we can answer questions such as whether a man is walking the dog, or whether the dog is chasing the man.

**Acknowledgements** This work was supported by the DARPA Transfer Learning program under contract number FA8750-05-2-0249 and the Multidisciplinary University Research Initiative (MURI), contract number N000140710747, managed by the Office of Naval Research. We would also like to thank Vittorio Ferrari and Pawan Kumar for providing us code and helping us to get their methods working on our data.

## Footnotes

*These authors contributed equally to this manuscript

## References

[1] D. Anguelov, P. Srinivasan, D. Koller, S. Thrun, J. Rodgers, and J. Davis. Scape: shape completion and animation of people. *SIGGRAPH*, '05. 3

[2] A. Bar-Hillel, T. Hertz, D. Weinshall. Efficient learning of relational object class models. *ICCV*, '05. 2

[3] A. Berg, T. Berg, and J. Malik. Shape matching and object recognition using low distortion correspondence. *CVPR*, '05. 2, 8

[4] T. Cootes, G. Edwards, and C. Taylor. Active appearance models. *ECCV*, '98. 1, 2, 3, 4

[5] G. Elidan, G. Heitz, and D. Koller. Learning object shape: From cartoons to images. *CVPR*, '06. 3

[6] G. Elidan, I. McGraw, and D. Koller. Residual belief propagation: Informed scheduling for async. message passing. *UAI*, '06. 4

[7] L. Fei-Fei and P. Perona. A bayesian hier. model for learning natural scene categories. *CVPR*, '05. 6

[8] L. Fei-Fei, R. Fergus, and P. Perona. Learning generative visual models from few training examples: an incremental bayesian approach tested on 101 object categories. *CVPR*, '04. 4, 7

[9] P. Felzenszwalb and D. Huttenlocher. Efficient matching of pictorial structures. *CVPR*, '00. 1

[10] P. Felzenszwalb and J. Schwartz. Hierarchical matching of deformable shapes. *CVPR*, '07.

[11] R. Fergus, P. Perona, and A. Zisserman. Object class recognition by unsupervised scale-invariant learning. *CVPR*, '03. 2

[12] V. Ferrari, F. Jurie, and C. Schmid. Accurate object detection with deformable shape models learnt from images. *CVPR*, '07. 2, 5, 8

[13] G. Heitz, G. Elidan, B. Packer, and D. Koller. Shape-based object localization for descriptive classification. Technical report, available at http://ai.stanford.edu/˜gaheitz/Research/Loops/TR.pdf 4

[14] A. Hill and C. Taylor. Non-rigid corresp. for automatic landmark identification. *BMVC*, '96. 2

[15] A. Holub and P. Perona. A discriminative framework for modeling object class. *CVPR*, '05.

[16] D. Lowe. Distinctive image features from scale-invariant keypoints. *IJCV*, '03. 3, 4

[17] A. Opelt, A. Pinz, and A. Zisserman. Incremental learning of object detectors using a visual shape alphabet. *CVPR*, '06. 2, 3, 8

[18] J. Pearl. *Probabilistic Reasoning in Intelligent Systems*. Morgan Kaufmann, '88. 3

[19] M. Prasad and A. Fitzgibbon. Single view reconstruction of curved surfaces. *CVPR 2006*, '06. 5, 8

[20] J. Sethian. *Level Set Methods and Fast Marching Methods*. Cambridge, '98. 2

[21] J. Shotton, A. Blake, and R. Cipolla. Contour-based learning for object detection. *ICCV*, '05. 2, 8

[22] A. Torralba, K. Murphy, and W. Freeman. Contextual models for object detection using boosted random fields. *NIPS*, '05. 3, 6

